# Crowdclustering

**Ryan Gomes**[*]     **Peter Welinder**     **Andreas Krause**     **Pietro Perona**
Caltech            Caltech         ETH Zurich & Caltech        Caltech

## Abstract

Is it possible to crowdsource categorization? Amongst the challenges: (a) each worker has only a partial view of the data, (b) different workers may have different clustering criteria and may produce different numbers of categories, (c) the underlying category structure may be hierarchical. We propose a Bayesian model of how workers may approach clustering and show how one may infer clusters / categories, as well as worker parameters, using this model. Our experiments, carried out on large collections of images, suggest that Bayesian crowdclustering works well and may be superior to single-expert annotations.

## 1   Introduction

Outsourcing information processing to large groups of anonymous workers has been made easier by the internet. *Crowdsourcing* services, such as Amazon's Mechanical Turk, provide a convenient way to purchase *Human Intelligence Tasks* (HITs). Machine vision and machine learning researchers have begun using crowdsourcing to label large sets of data (e.g., images and video [1, 2, 3]) which may then be used as training data for AI and computer vision systems. In all the work so far *categories* are defined by a scientist, while *categorical labels* are provided by the workers.

Can we use crowdsourcing to *discover* categories? I.e., is it possible to use crowdsourcing not only to *classify* data instances into established categories, but also to *define the categories* in the first place? This question is motivated by practical considerations. If we have a large number of images, perhaps several tens of thousands or more, it may not be realistic to expect a single person to look at all images and form an opinion as to how to categorize them. Additionally, individuals, whether untrained or expert, might not agree on the criteria used to define categories and may not even agree on the number of categories that are present. In some domains unsupervised clustering by machine may be of great help; however, unsupervised categorization of images and video is unfortunately a problem that is far from solved. Thus, it is an interesting question whether it is possible to collect and combine the opinion of multiple human operators, each one of which is able to view a (perhaps small) subset of a large image collection.

We explore the question of crowdsourcing clustering in two steps: (a) Reduce the problem to a number of independent HITs of reasonable size and assign them to a large pool of human workers (Section 2). (b) Develop a model of the annotation process, and use the model to aggregate the human data automatically (Section 3) yielding a partition of the dataset into categories. We explore the properties of our approach and algorithms on a number of real world data sets, and compare against existing methods in Section 4.

## 2   Eliciting Information from Workers

How shall we enable human operators to express their opinion on how to categorize a large collection of images? Whatever method we choose, it should be easy to learn and it should be implementable by means of a simple graphical user interface (GUI). Our approach (Figure 1) is based on displaying small subsets of $M$ images and asking workers to group them by means of mouse clicks. We provide instructions that may cue workers to certain attributes but we do not provide the worker with category definitions or examples. The worker groups the $M$ items into clusters of his choosing, as many as he sees fit. An item may be placed in its own cluster if it is unlike the others in the HIT. The choice of $M$ trades off between the difficulty of the task (worker time required for a HIT

---

[*]Corresponding author, e-mail: `gomes@vision.caltech.edu`

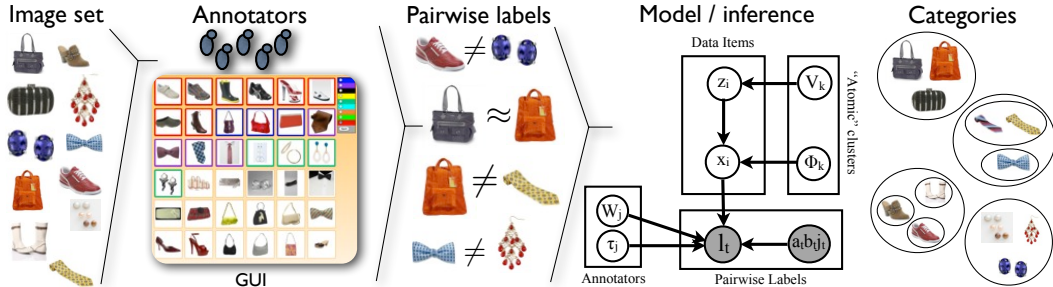

Figure 1: Schematic of Bayesian crowdclustering. A large image collection is explored by workers. In each HIT (Section 2), the worker views a small subset of images on a GUI. By associating (arbitrarily chosen) colors with sets of images the worker proposes a (partial) local clustering. Each HIT thus produces multiple binary pairwise labels: each pair of images shown in the same HIT is placed by the worker either in the same category or in different categories. Each image is viewed by multiple workers in different contexts. A model of the annotation process (Sec. 3.1) is used to compute the most likely set of categories from the binary labels. Worker parameters are estimated as well.

increases super-linearly with the number of items), the resolution of the images (more images on the screen means that they will be smaller), and contextual information that may guide the worker to make more global category decisions (more images give a better context, see Section 4.1.) Partial clusterings on many $M$-sized subsets of the data from many different workers are thus the raw data on which we compute clustering.

An alternative would have been to use pairwise distance judgments or three-way comparisons. A large body of work exists in the social sciences that makes use of human-provided similarity values defined between pairs of data items (e.g., Multidimensional Scaling [4].) After obtaining pairwise similarity ratings from workers, and producing a Euclidean embedding, one could conceivably proceed with unsupervised clustering of the data in the Euclidean space. However, accurate distance judgments may be more laborious to specify than partial clusterings. We chose to explore what we can achieve with partial clusterings alone.

We do not expect workers to agree on their definitions of categories, or to be consistent in categorization when performing multiple HITs. Thus, we avoid explicitly associating categories across HITs. Instead, we represent the results of each HIT as a series of $\binom{M}{2}$ binary labels (see Figure 1). We assume that there are $N$ total items (indexed by $i$), $J$ workers (indexed by $j$), and $H$ HITs (indexed by $h$). The information obtained from workers is a set of binary variables $\mathcal{L}$, with elements $l_t \in \{-1, +1\}$ indexed by a positive integer $t \in \{1, \ldots, T\}$. Associated with the $t$-th label is a quadruple $(a_t, b_t, j_t, h_t)$, where $j_t \in \{1, \ldots, J\}$ indicates the worker that produced the label, and $a_t \in \{1, \ldots, N\}$ and $b_t \in \{1, \ldots, N\}$ indicate the two data items compared by the label. $h_t \in \{1, \ldots, H\}$ indicates the HIT from which the $t$-th pairwise label was derived. The number of labels is $T = H\binom{M}{2}$.

**Sampling Procedure** We have chosen to structure HITs as clustering tasks of $M$ data items, so we must specify them. If we simply seperate the items into disjoint sets, then it will be impossible to infer a clustering over the entire data set. We will not know whether two items in different HITs are in the same cluster or not. There must be some overlap or redundancy: data items must be members of multiple HITs.

In the other extreme, we could construct HITs such that each pair of items may be found in at least one HIT, so that every possible pairwise category relation is sampled. This would be quite expensive for large number of items $N$, since the number of labels scales asymptotically as $T \in \Omega(N^2)$. However, we expect a noisy transitive property to hold: if items $a$ and $b$ are likely to be in the same cluster, and items $b$ and $c$ are (not) likely in the same cluster, then items $a$ and $c$ are (not) likely to be in the same cluster as well. The transitive nature of binary cluster relations should allow sparse sampling, especially when the number of clusters is relatively small.

As a baseline sampling method, we use the random sampling scheme outlined by Strehl and Ghosh [5] developed for the problem of *object distributed* clustering, in which a partition of a complete data set is learned from a number of clusterings restricted to subsets of the data. (We compare our aggregation algorithm to this work in Section 4.) Their scheme controls the level of sampling redundancy with a single parameter $V$, which in our problem is interpreted as the expected number of HITs to which a data item belongs.

The $N$ items are first distributed deterministically among the HITs, so that there are $\lceil \frac{M}{V} \rceil$ items in each HIT. Then the remaining $M - \lceil \frac{M}{V} \rceil$ items in each HIT are filled by sampling without replacement from the $N - \lceil \frac{M}{V} \rceil$ items that are not yet allocated to the HIT. There are a total of $\lceil \frac{NV}{M} \rceil$ unique HITs. We introduce an additional parameter $R$, which is the number of different workers that perform each constructed HIT. The total number of HITs distributed to the crowdsourcing service is therefore $H = R\lceil \frac{NV}{M} \rceil$, and we impose the constraint that a worker can not perform the same HIT more than once. This sampling scheme generates $T = R\lceil \frac{NV}{M} \rceil \binom{M}{2} \in O(RNVM)$ binary labels.

With this exception, we find a dearth of ideas in the literature pertaining to sampling methods for distributed clustering problems. Iterative schemes that adaptively choose maximally informative HITs may be preferable to random sampling. We are currently exploring ideas in this direction.

## 3   Aggregation via Bayesian Crowdclustering

There is an extensive literature in machine learning on the problem of combining multiple alternative clusterings of data. This problem is known as consensus clustering [6], clustering aggregation [7], or cluster ensembles [5]. While some of these methods can work with partial input clusterings, most have not been demonstrated in situations where the input clusterings involve only a small subset of the total data items ($M << N$), which is the case in our problem.

In addition, existing approaches focus on producing a single "average" clustering from a set of input clusterings. In contrast, we are not merely interested in the average clustering produced by a crowd of workers. Instead, we are interested in understanding the ways in which different individuals may categorize the data. We seek a master clustering of the data that may be combined in order to describe the tendencies of individual workers. We refer to these groups of data as *atomic* clusters.

For example, suppose one worker groups objects into a cluster of tall objects and another of short objects, while a different worker groups the same objects into a cluster of red objects and another of blue objects. Then, our method should recover four atomic clusters: tall red objects, short red objects, tall blue objects, and short blue objects. The behavior of the two workers may then be summarized using a confusion table of the atomic clusters (see Section 3.3). The first worker groups the first and third atomic cluster into one category and the second and fourth atomic cluster into another category. The second worker groups the first and second atomic clusters into a category and the third and fourth atomic clusters into another category.

### 3.1   Generative Model

We propose an approach in which data items are represented as points in a Euclidean space and workers are modeled as pairwise binary classifiers in this space. Atomic clusters are then obtained by clustering these inferred points using a Dirichlet process mixture model, which estimates the number of clusters [8]. The advantage of an intermediate Euclidean representation is that it provides a compact way to capture the characteristics of each data item. Certain items may be inherently more difficult to categorize, in which case they may lie between clusters. Items may be similar along one axis but different along another (e.g., object height versus object color.) A similar approach was proposed by Welinder et al. [3] for the analysis of classification labels obtained from crowdsourcing services. This method does not apply to our problem, since it involves binary labels applied to single data items rather than to pairs, and therefore requires that categories be defined a priori and agreed upon by all workers, which is incompatible with the crowdclustering problem.

We propose a probabilistic latent variable model that relates pairwise binary labels to hidden variables associated with both workers and images. The graphical model is shown in Figure 1. $\mathbf{x}_i$ is a $D$ dimensional vector, with components $[\mathbf{x}_i]_d$ that encodes item $i$'s location in the embedding space $\mathbb{R}^D$. Symmetric matrix $\mathbf{W}_j \in \mathbb{R}^{D \times D}$ with entries $[\mathbf{W}_j]_{d_1 d_2}$ and bias $\tau_j \in \mathbb{R}$ are used to define a pairwise binary classifier, explained in the next paragraph, that represents worker $j$'s labeling behavior. Because $\mathbf{W}_j$ is symmetric, we need only specify its upper triangular portion: vecp$\{\mathbf{W}_j\}$ which is a vector formed by "stacking" the partial columns of $\mathbf{W}_j$ according to the ordering $[\text{vecp}\{\mathbf{W}_j\}]_1 = [\mathbf{W}_j]_{11}, [\text{vecp}\{\mathbf{W}_j\}]_2 = [\mathbf{W}_j]_{12}, [\text{vecp}\{\mathbf{W}_j\}]_3 = [\mathbf{W}_j]_{22}$, etc. $\mathbf{\Phi}_k = \{\boldsymbol{\mu}_k, \boldsymbol{\Sigma}_k\}$ are the mean and covariance parameters associated with the $k$-th Gaussian atomic cluster, and $U_k$ are stick breaking weights associated with a Dirichlet process.

The key term is the pairwise quadratic logistic regression likelihood that captures worker $j$'s tendency to label the pair of images $a_t$ and $b_t$ with $l_t$:

$$p(l_t|\mathbf{x}_{a_t}, \mathbf{x}_{b_t}, \mathbf{W}_{j_t}, \tau_{j_t}) = \frac{1}{1 + \exp(-l_t A_t)} \tag{1}$$

where we define the pairwise quadratic *activity* $A_t = \mathbf{x}_{a_t}^T \mathbf{W}_{j_t} \mathbf{x}_{b_t} + \tau_{j_t}$. Symmetry of $\mathbf{W}_j$ ensures that $p(l_t|\mathbf{x}_{a_t}, \mathbf{x}_{b_t}, \mathbf{W}_{j_t}, \tau_{j_t}) = p(l_t|\mathbf{x}_{b_t}, \mathbf{x}_{a_t}, \mathbf{W}_{j_t}, \tau_{j_t})$. This form of likelihood yields a compact and tractable method of representing classifiers defined over pairs of points in Euclidean space. Pairs of vectors with large pairwise activity tend to be classified as being in the same category, and in different categories otherwise. We find that this form of likelihood leads to tightly grouped clusters of points $\mathbf{x}_i$ that are then easily discovered by mixture model clustering.

The joint distribution is

$$p(\Phi, U, Z, X, W, \tau, \mathcal{L}) = \prod_{k=1}^{\infty} p(U_k|\alpha)p(\Phi_k|\mathbf{m}_0, \beta_0, \mathbf{J}_0, \eta_0) \prod_{i=1}^{N} p(z_i|U)p(\mathbf{x}_i|\Phi_{z_i}) \qquad (2)$$

$$\prod_{j=1}^{J} p(\text{vecp}\{\mathbf{W}_j\}|\sigma_0^w)p(\tau_j|\sigma_0^\tau) \prod_{t=1}^{T} p(l_t|\mathbf{x}_{a_t}, \mathbf{x}_{b_t}, \mathbf{W}_{j_t}, \tau_{j_t}).$$

The conditional distributions are defined as follows:

$$p(U_k|\alpha) = \text{Beta}(U_k; 1, \alpha) \qquad\qquad p(z_i = k|U) = U_k \prod_{l=1}^{k-1}(1 - U_l) \quad (3)$$

$$p(\mathbf{x}_i|\Phi_{z_i}) = \text{Normal}(\mathbf{x}_i; \mu_{z_i}, \mathbf{\Sigma}_{z_i}) \qquad\qquad p(\mathbf{x}_i|\sigma_0^x) = \prod_d \text{Normal}([\mathbf{x}_i]_d; 0, \sigma_0^x)$$

$$p(\text{vecp}\{\mathbf{W}_j\}|\sigma_0^w) = \prod_{d_1 \le d_2} \text{Normal}([\mathbf{W}_j]_{d_1 d_2}; 0, \sigma_0^w) \qquad p(\tau_j|\sigma_0^\tau) = \text{Normal}(\tau_j; 0, \sigma_0^\tau)$$

$$p(\Phi_k|\mathbf{m}_0, \beta_0, \mathbf{J}_0, \eta_0) = \text{Normal-Wishart}(\Phi_k; \mathbf{m}_0, \beta_0, \mathbf{J}_0, \eta_0)$$

where $(\sigma_0^x, \sigma_0^\tau, \sigma_0^w, \alpha, \mathbf{m}_0, \beta_0, \mathbf{J}_0, \eta_0)$ are fixed hyper-parameters. Our model is similar to that of [9], which is used to model binary relational data. Salient differences include our use of a logistic rather than a Gaussian likelihood, and our enforcement of the symmetry of $\mathbf{W}_j$. In the next section, we develop an efficient deterministic inference algorithm to accomodate much larger data sets than the sampling algorithm used in [9].

### 3.2 Approximate Inference

Exact posterior inference in this model is intractable, since computing it involves integrating over variables with complex dependencies. We therefore develop an inference algorithm based on the Variational Bayes method [10]. The high level idea is to work with a factorized proxy posterior distribution that does not model the full complexity of interactions between variables; it instead represents a single mode of the true posterior. Because this distribution is factorized, integrations involving it become tractable. We define the proxy distribution $q(\Phi, U, Z, X, W, \tau) =$

$$\prod_{k=K+1}^{\infty} p(U_k|\alpha)p(\Phi_k|\mathbf{m}_0, \beta_0, \mathbf{J}_0, \eta_0) \prod_{k=1}^{K} q(U_k)q(\Phi_k) \prod_{i=1}^{N} q(z_i)q(\mathbf{x}_i) \prod_{j=1}^{J} q(\text{vecp}\{\mathbf{W}_j\})q(\tau_j) \quad (4)$$

using parametric distributions of the following form:

$$q(U_k) = \text{Beta}(U_k; \xi_{k,1}, \xi_{k,2}) \qquad\qquad q(\Phi_k) = \text{Normal-Wishart}(\mathbf{m}_k, \beta_k, \mathbf{J}_k, \eta_k) \quad (5)$$

$$q(\mathbf{x}_i) = \prod_d \text{Normal}([\mathbf{x}_i]_d; [\boldsymbol{\mu}_i^x]_d, [\boldsymbol{\sigma}_i^x]_d) \qquad\qquad q(\tau_j) = \text{Normal}(\tau_j; \mu_j^\tau, \sigma_j^\tau)$$

$$q(z_i = k) = q_{ik} \qquad\qquad q(\text{vecp}\{\mathbf{W}_j\}) = \prod_{d_1 \le d_2} \text{Normal}([\mathbf{W}_j]_{d_1 d_2}; [\boldsymbol{\mu}_j^w]_{d_1 d_2}, [\boldsymbol{\sigma}_j^w]_{d_1 d_2})$$

To handle the infinite number of mixture components, we follow the approach of [11] where we define variational distributions for the first $K$ components, and fix the remainder to their corresponding priors. $\{\xi_{k,1}, \xi_{k,2}\}$ and $\{\mathbf{m}_k, \beta_k, \mathbf{J}_k, \eta_k\}$ are the variational parameters associated with the $k$-th mixture component. $q(z_i = k) = q_{ik}$ form the factorized assignment distribution for item $i$. $\boldsymbol{\mu}_i^x$ and $\boldsymbol{\sigma}_i^x$ are variational mean and variance parameters associated with data item $i$'s embedding location. $\boldsymbol{\mu}_j^w$ and $\boldsymbol{\sigma}_j^w$ are symmetric matrix variational mean and variance parameters associated with worker $j$, and $\mu_j^\tau$ and $\sigma_j^\tau$ are variational mean and variance parameters for the bias $\tau_j$ of worker $j$. We use diagonal covariance Normal distributions over $\mathbf{W}_j$ and $\mathbf{x}_i$ to reduce the number of parameters that must be estimated.

Next, we define a utility function which allows us to determine the variational parameters. We use Jensen's inequality to develop a lower bound to the log evidence:

$$\log p(\mathcal{L}|\sigma_0^x, \sigma_0^\tau, \sigma_0^w, \alpha, \mathbf{m}_0, \beta_0, \mathbf{J}_0, \eta_0) \tag{6}$$
$$\geq E_q \log p(\Phi, U, Z, X, W, \tau, \mathcal{L}) + \mathcal{H}\{q(\Phi, U, Z, X, W, \tau)\},$$

$\mathcal{H}\{\cdot\}$ is the entropy of the proxy distribution, and the lower bound is known as the *Free Energy*. However, the Free Energy still involves intractable integration, because the normal distributions over variables $\mathbf{W}_j$, $\mathbf{x}_i$, and $\tau_j$ are not conjugate [12] to the logistic likelihood term. We therefore locally approximate the logistic likelihood with an unnormalized Gaussian function lower bound, which is the left hand side of the following inequality:

$$g(\Delta_t) \exp\{(l_t A_t - \Delta_t)/2 + \lambda(\Delta_t)(A_t^2 - \Delta_t^2)\} \leq p(l_t|\mathbf{x}_{a_t}, \mathbf{x}_{b_t}, \mathbf{W}_{j_t}, \tau_{j_t}). \tag{7}$$

This was adapted from [13] to our case of quadratic pairwise logistic regression. Here $g(x) = (1 + e^{-x})^{-1}$ and $\lambda(\Delta) = [1/2 - g(\Delta)]/(2\Delta)$. This expression introduces an additional variational parameter $\Delta_t$ for each label, which are optimized in order to tighten the lower bound. Our utility function is therefore:

$$\mathcal{F} = E_q \log p(\Phi, U, Z, X, W, \tau) + \mathcal{H}\{q(\Phi, U, Z, X, W, \tau)\} \tag{8}$$
$$+ \sum_t \log g(\Delta_t) + \frac{l_t}{2} E_q\{A_t\} - \frac{\Delta_t}{2} + \lambda(\Delta_t)(E_q\{A_t^2\} - \Delta_t^2)$$

which is a tractable lower bound to the log evidence. Optimization of variational parameters is carried out in a coordinate ascent procedure, which exactly maximizes each variational parameter in turn while holding all others fixed. This is guaranteed to converge to a local maximum of the utility function. The update equations are given in an extended technical report [14]. We initialize the variational parameters by carrying out a layerwise procedure: first, we substitute a zero mean isotropic normal prior for the mixture model and perform variational updates over $\{\boldsymbol{\mu}_i^x, \boldsymbol{\sigma}_i^x, \boldsymbol{\mu}_j^w, \boldsymbol{\sigma}_j^w, \mu_j^\tau, \sigma_j^\tau\}$. Then we use $\boldsymbol{\mu}_i^x$ as point estimates for $\mathbf{x}_i$ and update $\{\mathbf{m}_k, \beta_k, \mathbf{J}_k, \eta_k, \xi_{k,1}, \xi_{k,2}\}$ and determine the initial number of clusters $K$ as in [11]. Finally, full joint inference updates are performed. Their computational complexity is $O(D^4 T + D^2 KN) = O(D^4 NVRM + D^2 KN)$.

### 3.3 Worker Confusion Analysis

As discussed in Section 3, we propose to understand a worker's behavior in terms of how he groups atomic clusters into his own notion of categories. We are interested in the predicted confusion matrix $\mathbf{C}_j$ for worker $j$, where

$$[\mathbf{C}_j]_{k_1 k_2} = E_q\left\{ \int p(l = 1|\mathbf{x}_a, \mathbf{x}_b, \mathbf{W}_j, \tau_j) p(\mathbf{x}_a|\Phi_{k_1}) p(\mathbf{x}_b|\Phi_{k_2}) d\mathbf{x}_a d\mathbf{x}_b \right\} \tag{9}$$

which expresses the probability that worker $j$ assigns data items sampled from atomic cluster $k_1$ and $k_2$ to the same cluster, as predicted by the variational posterior. This integration is intractable. We use the expected values $E\{\Phi_{k_1}\} = \{\mathbf{m}_{k_1}, \mathbf{J}_{k_1}/\eta_{k_1}\}$ and $E\{\Phi_{k_2}\} = \{\mathbf{m}_{k_2}, \mathbf{J}_{k_2}/\eta_{k_2}\}$ as point estimates in place of the variational distributions over $\Phi_{k_1}$ and $\Phi_{k_2}$. We then use Jensen's inequality and Eq. 7 again to yield a lower bound. Maximizing this bound over $\Delta$ yields

$$[\hat{\mathbf{C}}_j]_{k_1 k_2} = g(\hat{\Delta}_{k_1 k_2 j}) \exp\{(\mathbf{m}_{k_1}^T \boldsymbol{\mu}_j^w \mathbf{m}_{k_2} + \mu_j^\tau - \hat{\Delta}_{k_1 k_2 j})/2\} \tag{10}$$

which we use as our approximate confusion matrix, where $\hat{\Delta}_{k_1 k_2 j}$ is given in [14].

## 4 Experiments

We tested our method on four image data sets that have established "ground truth" categories, which were provided by a single human expert. These categories do not necessarily reflect the uniquely valid way to categorize the data set, however they form a convenient baseline for the purpose of quantitative comparison. We used 1000 images from the Scenes data set from [15] to illustrate our approach (Figures 2, 3, and 4.) We used 1354 images of birds from 10 species in the CUB-200 data set [16] (Table 1) and the 3845 images in the Stonefly9 data set [17] (Table 1) in order to compare our method quantitatively to other cluster aggregation methods. We used the 37794 images from the Attribute Discovery data set [18] in order to demonstrate our method on a large scale problem.

We set the dimensionality of $\mathbf{x}_i$ to $D = 4$ (since higher dimensionality yielded no additional clusters) and we iterated the update equations 100 times, which was enough for convergence. Hyperparameters were tuned once on synthetic pairwise labels that simulated 100 data points drawn from 4 clusters, and fixed during all experiments.

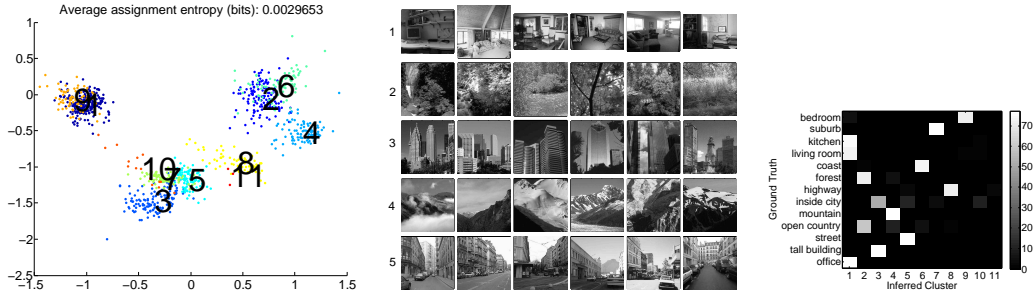

Figure 2: Scene Dataset. Left: Mean locations $\boldsymbol{\mu}_i^x$ projected onto first two Fisher discriminant vectors, along with cluster labels superimposed at cluster means $\mathbf{m}_k$. Data items are colored according to their MAP label $\mathrm{argmax}_k\, q_{ik}$. Center: High confidence example images from the largest five clusters (rows correspond to clusters.) Right: Confusion table between ground truth scene categories and inferred clusters. The first cluster includes three indoor ground truth categories, the second includes *forest* and *open country* categories, and the third includes two urban categories. See Section 4.1 for a discussion and potential solution of this issue.

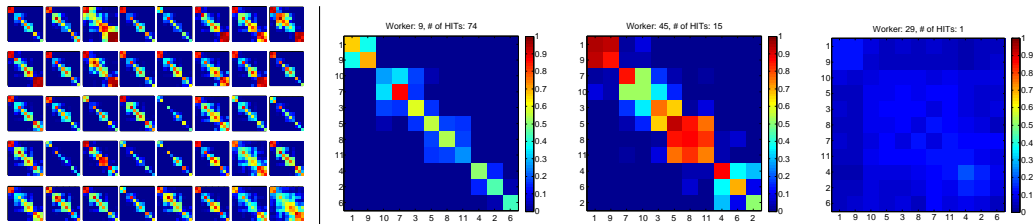

Figure 3: (Left of line) Worker confusion matrices for the 40 most active workers. (Right of line) Selected worker confusion matrices for Scenes experiment. Worker 9 (left) makes distinctions that correspond closely to the atomic clustering. Worker 45 (center) makes coarser distinctions, often combining atomic clusters. Right: Worker 29's single HIT was largely random and does not align with the atomic clusters.

Figure 2 (left) shows the mean locations of the data items $\boldsymbol{\mu}_i^x$ learned from the Scene data set, visualized as points in Euclidean space. We find well seperated clusters whose labels $k$ are displayed at their mean locations $\mathbf{m}_k$. The points are colored according to $\mathrm{argmax}_k\, q_{ik}$, which is item $i$'s MAP cluster assignment. The cluster labels are sorted according to the number of assigned items, with cluster 1 being the largest. The axes are the first two Fisher discriminant directions (derived from the MAP cluster assignments) as axes. The clusters are well seperated in the four dimensionsal space (we give the average assignment entropy $-\frac{1}{N}\sum_{ik} q_{ik}\log q_{ik}$ in the figure title, which shows little cluster overlap.) Figure 2 (center) shows six high confidence examples from clusters 1 through 5. Figure 2 (right) shows the confusion table between the ground truth categories and the MAP clustering. We find that the MAP clusters often correspond to single ground truth categories, but they sometimes combine ground truth categories in reasonable ways. See Section 4.1 for a discussion and potential solution of this issue.

Figure 3 (left of line) shows the predicted confusion matrices (Section 3.3) associated with the 40 workers that performed the most HITs. This matrix captures the worker's tendency to label items from different atomic clusters as being in the same or different category. Figure 3 (right of line) shows in detail the predicted confusion matrices for three workers. We have sorted the MAP cluster indices to yield approximately block diagonal matrices, for ease of interpretation. Worker 9 makes relatively fine grained distinctions, including seperating clusters 1 and 9 that correspond to the indoor categories and the bedroom scenes, respectively. Worker 45 combines clusters 5 and 8 which correspond to city street and highway scenes in addition to grouping together all indoor scene categories. The finer grained distinctions made by worker 9 may be a result of performing more HITs (74) and seeing a larger number of images than worker 45, who performed 15 HITs. Finally (far right), we find a worker whose labels do not align with the atomic clustering. Inspection of his labels show that they were entered largely at random.

Figure 4 (top left) shows the number of HITs performed by each worker according to descending rank. Figure 4 (bottom left) is a Pareto curve that indicates the percentage of the HITs performed by the most active workers. The Pareto principle (i.e., the law of the vital few) [19] roughly holds: the top 20% most active workers perform nearly 80% of the work. We wish to understand the extent to which the most active workers contribute to the results. For the purpose of quantitative comparisons, we use Variation of Information (VI) [20] to measure the discrepancy between the

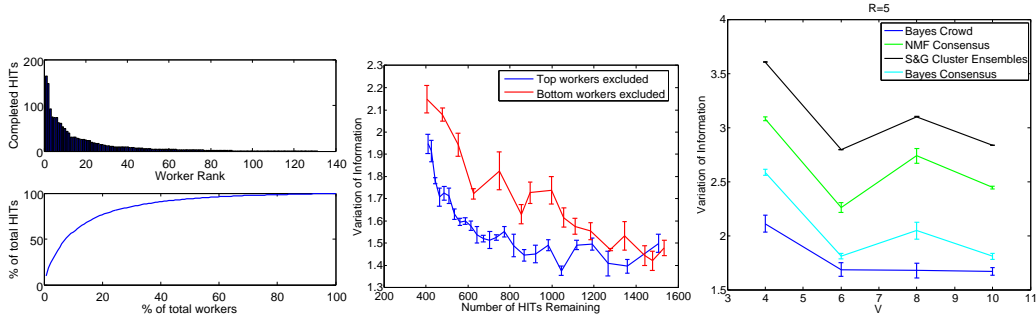

Figure 4: Scene Data set. Left Top: Number of completed HITs by worker rank. Left Bottom: Pareto curve. Center: Variation of Information on the Scene data set as we incrementally remove top (blue) and bottom (red) ranked workers. The top workers are removed one at a time, bottom ranked workers are removed in groups so that both curves cover roughly the same domain. The most active workers do not dominate the results. Right: Variation of Information between the inferred clustering and the ground truth categories on the Scene data set, as a function of sampling parameter $V$. $R$ is fixed at 5.

|  | Bayes Crowd | Bayes Consensus | NMF [21] | Strehl & Ghosh [5] |
|---|---|---|---|---|
| Birds [16] (VI) | **1.103 ± 0.082** | 1.721 ± 0.07 | 1.500 ± 0.26 | 1.256 ± 0.001 |
| Birds (time) | 18.5 min | 18.1 min | 27.9 min | **0.93** min |
| Stonefly9 [17] (VI) | **2.448 ± 0.063** | 2.735 ± 0.037 | 4.571 ± 0.158 | 3.836 ± 0.002 |
| Stonefly9 (time) | 100.1 min | 98.5 min | 212.6 min | **46.5** min |

Table 1: Quantitative comparison on Bird and Stonefly species categorization data sets. Quality is measured using Variation of Information between the inferred clustering and ground truth. Bayesian Crowdclustering outperforms the alternatives.

inferred MAP clustering and the ground truth categorization. VI is a metric with strong information theoretic justification that is defined between two partitions (clusterings) of a data set; smaller values indicate a closer match and a VI of 0 means that two clusterings are identical. In Figure 4 (center) we incrementally remove the most active (blue) and least active (red) workers. Removal of workers corresponds to moving from right to left on the x-axis, which indicates the number of HITs used to learn the model. The results show that removing the large number of workers that do fewer HITs is more detrimental to performance than removing the relatively few workers that do a large number of HITs (given the same number of total HITs), indicating that the atomic clustering is learned from the crowd at large.

In Figure 4 (right), we judge the impact of the sampling redundancy parameter $V$ described in Section 2. We compare our approach (Bayesian crowdclustering) to two existing clustering aggregation methods from the literature: consensus clustering by nonnegative matrix factorization (NMF) [21] and the cluster ensembles method of Strehl and Ghosh (S&G) [5]. NMF and S&G require the number of inferred clusters to be provided as a parameter, and we set this to the number of ground truth categories. Even without the benefit of this additional information, our method (which automatically infers the number of clusters) outperforms the alternatives. To judge the benefit of modeling the characteristics of individual workers, we also compare against a variant of our model in which all HITs are treated as if they are performed by a single worker (Bayesian consensus.) We find a significant improvement. We fix $R = 5$ in this experiment, but we find a similar ranking of methods at other values of $R$. However, the performance benefit of the Bayesian methods over the existing methods increases with $R$.

We compare the four methods quantitatively on two additional data sets, with the results summarized in Table 1. In both cases, we instruct workers to categorize based on species. This is known to be a difficult task for non-experts. We set $V = 6$ and $R = 5$ for these experiments. Again, we find that Bayesian Crowdclustering outperforms the alternatives. A run time comparison is also given in Table 1. Bayesian Crowdclustering results on the Bird and Stonefly data sets are summarized in [14].

Finally, we demonstrate Bayesian crowdclustering on the large scale Attribute Discovery data set. This data set has four image categories: bags, earrings, ties, and women's shoes. In addition, each image is a member of one of 27 sub-categories (e.g., the bags category includes backpacks and totes as sub-categories.) See [14] for summary figures. We find that our method easily discovers the four

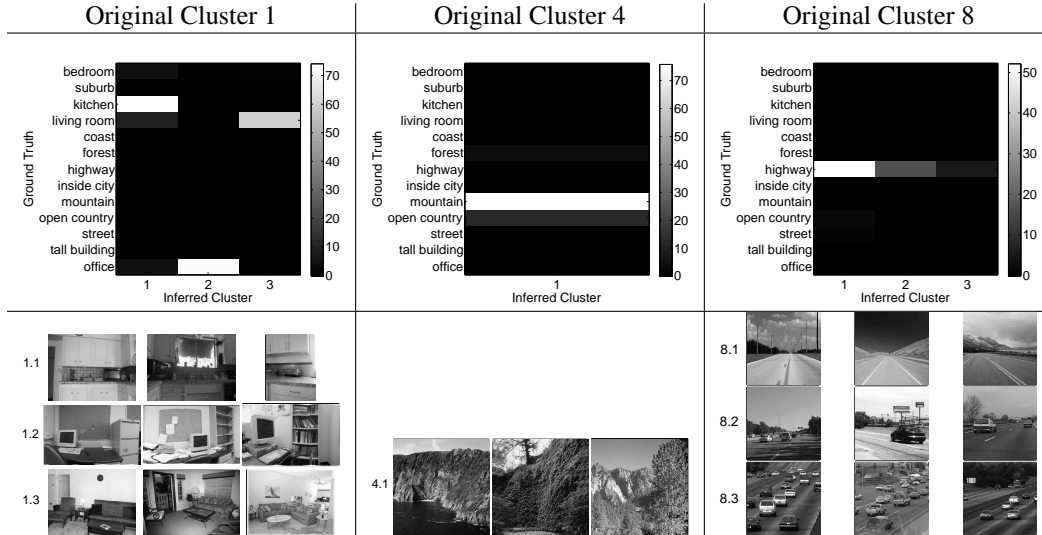

Figure 5: Divisive Clustering on the Scenes data set. Left: Confusion matrix and high confidence examples when running our method on images assigned to cluster one in the original experiment (Figure 2). The three indoor scene categories are correctly recovered. Center: Workers are unable to subdivide mountain scenes consistently and our method returns a single cluster. Right: Workers may find perceptually relevant distinctions not present in the ground truth categories. Here, the highway category is subdivided according to the number of cars present.

categories. The subcategories are not discovered, likely due to limited context associated with HITs with size $M = 36$ as discussed in the next section. Runtime was approximately 9.5 hours on a six core Intel Xeon machine.

## 4.1 Divisive Clustering

As indicated by the confusion matrix in Figure 2 (right), our method results in clusters that correspond to reasonable categories. However, it is clear that the data often has finer categorical distinctions that go undiscovered. We conjecture that this is a result of the limited context presented to the worker in each HIT. When shown a set of $M = 36$ images consisting mostly of different types of outdoor scenes and a few indoor scenes, it is reasonable for a worker to consider the indoor scenes as a unified category. However, if a HIT is composed purely of indoor scenes, a worker might draw finer distinctions between images of offices, kitchens, and living rooms. To test this conjecture, we developed a hierarchical procedure in which we run Bayesian crowdclustering independently on images that are MAP assigned to the same cluster in the original Scenes experiment.

Figure 5 (left) shows the results on the indoor scenes assigned to original cluster 1. We find that when restricted to indoor scenes, the workers do find the relevant distinctions and our algorithm accurately recovers the kitchen, living room, and office ground truth categories. In Figure 5 (center) we ran the procedure on images from original cluster 4, which is composed predominantly of mountain scenes. The algorithm discovers one subcluster. In Figure 5 (right) the workers divide a cluster into three subclusters that are perceptually relevant: they have organized them according to the number of cars present.

## 5 Conclusions

We have proposed a method for clustering a large set of data by distributing small tasks to a large group of workers. It is based on using a novel model of human clustering, as well as a novel machine learning method to aggregate worker annotations. Modeling both data item properties and the workers' annotation process and parameters appears to produce performance that is superior to existing clustering aggregation methods. Our study poses a number of interesting questions for further research: Can adaptive sampling methods (as opposed to our random sampling) reduce the number of HITs that are necessary to achieve high quality clustering? Is it possible to model the workers' tendency to learn over time as they perform HITs, rather than treating HITs independently as we do here? Can we model contextual effects, perhaps by modeling the way that humans "regularize" their categorical decisions depending on the number and variety of items present in the task?

**Acknowledgements**  This work was supported by ONR MURI grant 1015-G-NA-127, ARL grant W911NF-10-2-0016, and NSF grants IIS-0953413 and CNS-0932392.

# References

[1] A. Sorokin and D. A. Forsyth. Utility data annotation with Amazon Mechanical Turk. In *Internet Vision*, pages 1–8, 2008.

[2] Sudheendra Vijayanarasimhan and Kristen Grauman. Large-Scale Live Active Learning: Training Object Detectors with Crawled Data and Crowds. In *CVPR*, 2011.

[3] Peter Welinder, Steve Branson, Serge Belongie, and Pietro Perona. The multidimensional wisdom of crowds. In *Neural Information Processing Systems Conference (NIPS)*, 2010.

[4] J. B. Kruskal. Multidimensional scaling by optimizing goodness-of-fit to a nonmetric hypothesis. *PSym*, 29:1–29, 1964.

[5] Alexander Strehl and Joydeep Ghosh. Cluster ensembles—A knowledge reuse framework for combining multiple partitions. *Journal of Machine Learning Research*, 3:583–617, 2002.

[6] Stefano Monti, Pablo Tamayo, Jill Mesirov, and Todd Golub. Consensus clustering: A resampling-based method for class discovery and visualization of gene expression microarray data. *Machine Learning*, 52(1–2):91–118, 2003.

[7] Gionis, Mannila, and Tsaparas. Clustering aggregation. In *ACM Transactions on Knowledge Discovery from Data*, volume 1. 2007.

[8] A.Y. Lo. On a class of bayesian nonparametric estimates: I. density estimates. *The Annals of Statistics*, pages 351–357, 1984.

[9] I. Sutskever, R. Salakhutdinov, and J.B. Tenenbaum. Modelling relational data using bayesian clustered tensor factorization. *Advances in Neural Information Processing Systems (NIPS)*, 2009.

[10] Hagai Attias. A variational baysian framework for graphical models. In *NIPS*, pages 209–215, 1999.

[11] Kenichi Kurihara, Max Welling, and Nikos Vlassis. Accelerated variational dirichlet process mixtures. In B. Schölkopf, J. Platt, and T. Hoffman, editors, *Advances in Neural Information Processing Systems 19*. MIT Press, Cambridge, MA, 2007.

[12] J. M. Bernardo and A. F. M. Smith. *Bayesian Theory*. Wiley, 1994.

[13] Tommi S. Jaakkola and Michael I. Jordan. A variational approach to Bayesian logistic regression models and their extensions, August 13 1996.

[14] Ryan Gomes, Peter Welinder, Andreas Krause, and Pietro Perona. Crowdclustering. Technical Report CaltechAUTHORS:20110628-202526159, June 2011.

[15] Li Fei-Fei and Pietro Perona. A Bayesian hierarchical model for learning natural scene categories. In *CVPR*, pages 524–531. IEEE Computer Society, 2005.

[16] P. Welinder, S. Branson, T. Mita, C. Wah, F. Schroff, S. Belongie, and P. Perona. Caltech-UCSD Birds 200. Technical Report CNS-TR-2010-001, California Institute of Technology, 2010.

[17] G. Martinez-Munoz, N. Larios, E. Mortensen, W. Zhang, A. Yamamuro, R. Paasch, N. Payet, D. Lytle, L. Shapiro, S. Todorovic, et al. Dictionary-free categorization of very similar objects via stacked evidence trees. 2009.

[18] T. Berg, A. Berg, and J. Shih. Automatic attribute discovery and characterization from noisy web data. *Computer Vision–ECCV 2010*, pages 663–676, 2010.

[19] V. Pareto. Cours d'economie politique. 1896.

[20] M. Meila. Comparing clusterings by the variation of information. In *Learning theory and Kernel machines: 16th Annual Conference on Learning Theory and 7th Kernel Workshop, COLT/Kernel 2003, Washington, DC, USA, August 24-27, 2003: proceedings*, volume 2777, page 173. Springer Verlag, 2003.

[21] Tao Li, Chris H. Q. Ding, and Michael I. Jordan. Solving consensus and semi-supervised clustering problems using nonnegative matrix factorization. In *ICDM*, pages 577–582. IEEE Computer Society, 2007.

